# Using the Future to "Sort Out" the Present: Rankprop and Multitask Learning for Medical Risk Evaluation

**Rich Caruana, Shumeet Baluja, and Tom Mitchell**
School of Computer Science, Carnegie Mellon University, Pittsburgh, PA 15213
(caruana, baluja, mitchell)@cs.cmu.edu

## Abstract

A patient visits the doctor; the doctor reviews the patient's history, asks questions, makes basic measurements (blood pressure, ...), and prescribes tests or treatment. The prescribed course of action is based on an assessment of patient risk—patients at higher risk are given more and faster attention. It is also sequential—it is too expensive to immediately order all tests which might later be of value. This paper presents two methods that together improve the accuracy of backprop nets on a pneumonia risk assessment problem by 10-50%. *Rankprop* improves on backpropagation with sum of squares error in ranking patients by risk. *Multitask learning* takes advantage of *future* lab tests available in the training set, but not available in practice when predictions must be made. Both methods are broadly applicable.

## 1 Background

There are 3,000,000 cases of pneumonia each year in the U.S., 900,000 of which are admitted to the hospital for treatment and testing. Most pneumonia patients recover given appropriate treatment, and many can be treated effectively without hospitalization. Nonetheless, pneumonia is serious: 100,000 of those hospitalized for pneumonia die from it, and many more are at elevated risk if not hospitalized.

### 1.1 The Problem

A primary goal of medical decision making is to accurately, swiftly, and economically identify patients at high risk from diseases like pneumonia so they may be hospitalized to receive aggressive testing and treatment; patients at low risk may be more comfortably, safely, and economically treated at home. Note that the diagno-

sis of pneumonia has already been made; the goal is not to determine the illness, but how much risk the illness poses to the patient. Some of the most useful tests for doing this require hospitalization and will be available only if preliminary assessment indicates it is warranted. Low risk patients can safely be treated as outpatients and can often be identified using measurements made prior to admission.

The problem considered in this paper is to learn to rank pneumonia patients according to their probability of mortality. We present two learning methods that combined outperform standard backpropagation by 10-50% in identifying groups of patients with least mortality risk. These methods are applicable to domains where the goal is to rank instances according to a probability function and where useful attributes do not become available until after the prediction must be made. In addition to medical decision making, this class includes problems as diverse as investment analysis in financial markets and autonomous vehicle navigation.

## 1.2   The Pneumonia Database

The Medis Pneumonia Database [6] contains 14,199 pneumonia cases collected from 78 hospitals in 1989. Each patient in the database was diagnosed with pneumonia and hospitalized. 65 measurements are available for most patients. These include 30 basic measurements typically acquired prior to hospitalization such as age, sex, and pulse, and 35 lab results such as blood counts or gases not available until after hospitalization. The database indicates how long each patient was hospitalized and whether the patient lived or died. 1,542 (10.9%) of the patients died.

## 1.3   The Performance Criterion

The Medis database indicates which patients lived or died. The most useful decision aid for this problem would predict which patients will live or die. But this is too difficult. In practice, the best that can be achieved is to estimate a probability of death (POD) from the observed symptoms. In fact, it is sufficient to learn to *rank* patients by POD so lower risk patients can be discriminated from higher risk patients. The patients at least risk may then be considered for outpatient care.

The performance criterion used by others working with the Medis database [4] is the accuracy with which one can select a prespecified fraction of the patient population that do not die. For example, given a population of 10,000 patients, find the 20% of this population at *least* risk. To do this we learn a risk model and a threshold for this model that allows 20% of the population (2000 patients) to fall below it. If 30 of the 2000 patients below this threshold died, the error rate is $30/2000 = 0.015$. We say that the error rate for FOP 0.20 is 0.015 for this model ("FOP" stands for fraction of population). In this paper we consider FOPs 0.1, 0.2, 0.3, 0.4, and 0.5. Our goal is to learn models and model thresholds, such that the error rate at each FOP is minimized. Models with acceptably low error rates might then be employed to help determine which patients do not require hospitalization.

## 2   Methodology

The Medis database is unusually large, with over 14K training patterns. Because we are interested in developing methods that will be effective in other domains where databases of this size are not available, we perform our experiments using small training sets randomly drawn from the 14K patterns and use the remaining patterns as test sets. For each method we run ten trials. For each trial we randomly sample 2K patterns from the 14K pool for training. The 2K training sample is further split into a 1K backprop set used to train the net and a 1K halting set used to determine

when to halt training.[1] Once the network is trained, we run the 1K halt set through the model again to find the threshold that passes 10%, 20%, 30%, 40%, and 50% of the halt set. The performance of the model is evaluated on the 12K unused patterns by determining how many of the cases that fall below threshold in this test set die. This is the error rate for that model at that FOP.

## 3   The Traditional Approach: SSE on 0/1 Targets

Sections 3-5 present three neural net approaches to pneumonia risk prediction. This section presents the standard approach: using backpropagation on sum of squares errors (SSE) with 0=lives/1=dies to predict mortality. This works well if early stopping is used to prevent overfitting. Section 4 presents rankprop (SSE on ranks instead of 0/1 targets). Rankprop, which learns to rank patients by risk instead of directly predicting mortality, works better. Section 5 uses multitask learning (MTL) to benefit from tests in the database that in practice will not be available until after deciding to admit the patient. Rankprop with MTL works even better.

The straightforward approach to this problem is to use backprop to train a net to learn to predict which patients live or die, and then use the real-valued predictions of this net to sort patients by risk. This net has 30 inputs, 1 for each of the observed patient measurements, a hidden layer with 8 units[2], and a single output trained with 0=lived, 1=died.[3] Given an infinite training set, a net trained this way should learn to predict the probability of death for each patient, not which patients live or die. In the real world, however, where we rarely have an infinite number of training cases, a net will overtrain and begin to learn a very nonlinear function that outputs values near 0/1 for cases in the training set, but which does not generalize well. In this domain it is critical to use early stopping to halt training before this happens.

Table 1 shows the error rates of nets trained with SSE on 0/1 targets for the five FOPs. Each entry is the mean of ten trials. The first entry in the table indicates that on average, in the 10% of the test population predicted by the nets to be at least risk, 1.4% died. We do not know the best achievable error rates for this data.

Table 1: Error Rates of SSE on 0/1 Targets

| FOP | 0.1 | 0.2 | 0.3 | 0.4 | 0.5 |
|---|---|---|---|---|---|
| Error Rate | .0140 | .0190 | .0252 | .0340 | .0421 |

## 4   Using Rankprop to Rank Cases by Risk

Because the goal is to find the fraction of the population least likely to die, it is sufficient just to learn to rank patients by risk. Rankprop learns to rank patients without learning to predict mortality. "Rankprop" is short for "backpropagation using sum of squares errors on estimated ranks". The basic idea is to sort the training set using the target values, scale the ranks from this sort (we scale uniformly to [0.25,0.75] with sigmoid output units), and use the scaled ranks as target values for standard backprop with SSE instead of the 0/1 values in the database.

Ideally, we'd rank the training set by the true probabilities of death. Unfortunately, all we know is which patients lived or died. In the Medis database, 89% of the target values are 0's and 11% are 1's. There are many possible sorts consistent with these values. Which sort should backprop try to fit? It is the large number of possible sorts of the training set that makes backpropagating ranks challenging. Rankprop solves this problem by using the net model *as it is being learned* to order the training set *when target values are tied*. In this database, where there are many ties because there are only two target values, finding a proper ranking of the training set is a serious problem. Rankprop learns to adjust the target ranks *of* the training set at the same time it is learning to predict ranks *from* that training set.

How does rankprop do this? Rankprop alternates between rank passes and backprop passes. On the rank pass it records the output of the net for each training pattern. It then sorts the training patterns using the *target* values (0 or 1 in the Medis database), *but using the network's predictions for each pattern as a* **secondary** *sort key to break ties.* The basic idea is to find the legal rank of the target values (0 or 1) maximally consistent with the ranks the current model predicts. This *closest match* ranking of the target values is then used to define the target ranks used on the next backprop pass through the training set. Rankprop's pseudo code is:

```
foreach epoch do {
   foreach pattern do {
      network_output[pattern] = forward_pass(pattern)}
   target_rank = sort_and_scale_patterns(target_value, network_output)
   foreach pattern do {
      backprop(target_rank[pattern] - network_output[pattern])}}
```

where "sort_and_scale_patterns" sorts and ranks the training patterns using the sort keys specified in its arguments, the second being used to break ties in the first.

Table 2 shows the mean rankprop performance using nets with 8 hidden units. The bottom row shows improvements over SSE on 0/1 targets. All differences are statistically significant. See Section 7.1 for discussion of why rankprop works better.

Table 2: Error Rates of Rankprop and Improvement Over Standard Backprop

| FOP | 0.1 | 0.2 | 0.3 | 0.4 | 0.5 |
|---|---|---|---|---|---|
| Error Rate | .0083 | .0144 | .0210 | .0289 | .0386 |
| % Change | -40.7% | -24.2% | -16.7% | -15.0% | -8.3% |

## 5   Learning From the Future with Multitask Learning

The Medis database contains results from 36 lab tests that will be available only after patients are hospitalized. Unfortunately, these results will not be available when the model is used because the patients will not yet have been admitted. Multitask learning (MTL) improves generalization by having a learner simultaneously learn sets of related tasks with a shared representation; what is learned for each task might benefit other tasks. In this application, we use MTL to benefit from the future lab results. The extra lab values are used as extra backprop *outputs* as shown in Figure 1. The extra outputs bias the shared hidden layer towards representations that better capture important features of the domain. See [2][3][9] for details about MTL and [1] for other ways of using extra outputs to bias learning.

The MTL net has 64 hidden units. Table 3 shows the mean performance of ten runs of MTL with rankprop. The bottom row shows the improvement over rankprop

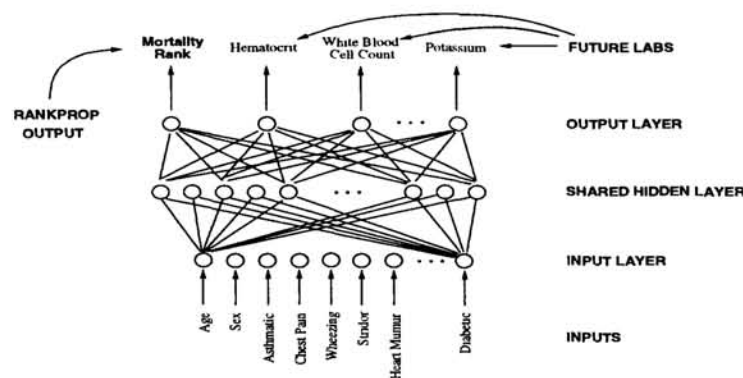

Figure 1: Using Future Lab Results as Extra Outputs To Bias Learning

alone. Although MTL lowers error at each FOP, only the differences at FOP = 0.3, 0.4, and 0.5 are statistically significant with ten trials. Feature nets [7], a competing approach that trains nets to predict the missing future labs and uses the predictions as extra net *inputs* does not yield benefits comparable to MTL on this problem.

Table 3: Error Rates of Rankprop+MTL and Improvement Over Rankprop Alone

| FOP | 0.1 | 0.2 | 0.3 | 0.4 | 0.5 |
|---|---|---|---|---|---|
| Error Rate | .0074 | .0127 | .0197 | .0269 | .0364 |
| % Change | -10.8% | -11.8% | -6.2% | -6.9% | -5.7% |

## 6   Comparison of Results

Table 4 compares the performance of backprop using SSE on 0/1 targets with the combination of rankprop and multitask learning. On average, Rankprop+MTL reduces error more than 25%. This improvement is not easy to achieve—experiments with other learning methods such as Bayes Nets, Hierarchical Mixtures of Experts, and K-Nearest Neighbor (run not by us, but by experts in their use) indicate SSE on 0/1 targets is an excellent performer on this domain[4].

Table 4: Comparison Between SSE on 0/1 Targets and Rankprop+MTL

| FOP | 0.1 | 0.2 | 0.3 | 0.4 | 0.5 |
|---|---|---|---|---|---|
| SSE on 0/1 | .0140 | .0190 | .0252 | .0340 | .0421 |
| Rankprop+MTL | .0074 | .0127 | .0197 | .0269 | .0364 |
| % Change | -47.1% | -33.2% | -21.8% | -20.9% | -13.5% |

## 7   Discussion

### 7.1   Why Does Rankprop Work?

We are given data from a target function $f(x)$. Suppose the goal is not to learn a model *of* $f(x)$, but to learn to sort patterns *by* $f(x)$. Must we learn a model of $f(x)$ and use its predictions for sorting? No. It suffices to learn a function $g(x)$ such that for all $x_1, x_2$, $[g(x_1) \le g(x_2)] \rightarrow [f(x_1) \le f(x_2)]$. There can be many such functions $g(x)$ for a given $f(x)$, and some of these may be easier to learn than $f(x)$.

Consider the probability function in Figure 2.1 that assigns to each $x$ the probability $p = f(x)$ that the outcome is 1; with probability $1 - p$ the outcome is 0. Figure 2.2 shows a training set sampled from this distribution. Where the probability is low, there are many 0's. Where the probability is high, there are many 1's. Where the probability is near 0.5, there are 0's and 1's. *This region causes problems for backprop using SSE on 0/1 targets: similar inputs are mapped to dissimilar targets.*

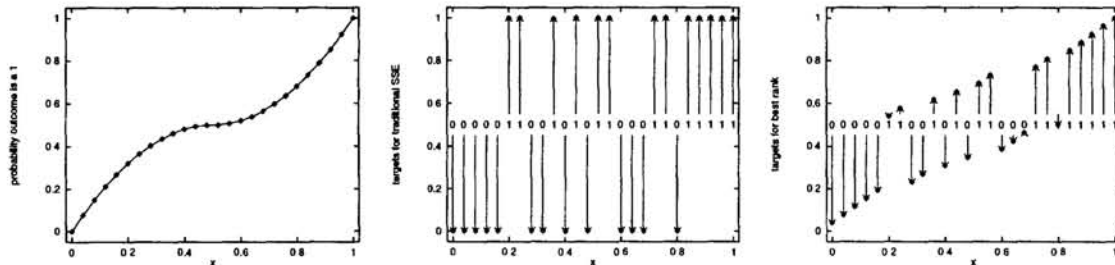

Figure 2: SSE on 0/1 Targets and on Ranks for a Simple Probability Function

Backprop learns a very nonlinear function if trained on Figure 2.2. This is unfortunate: Figure 2.1 is smooth and maps similar inputs to similar outputs. If the goal is to learn to rank the data, we can learn a simpler, *less nonlinear* function instead. There exists a ranking of the training data such that if the ranks are used as backprop target values, the resulting function is less nonlinear than the original target function. Figure 2.3 shows these target rank values. Similar input patterns have more similar rank target values than the original target values.

Rankprop tries to learn *simple* functions that directly support ranking. One difficulty with this is that rankprop must learn a ranking of the training data while also training the model to predict ranks. We do not yet know under what conditions this parallel search will converge. We conjecture that when rankprop does converge, it will often be to simpler models than it would have learned from the original target values (0/1 in Medis), and that these simpler models will often generalize better.

## 7.2   Other Applications of Rankprop and Learning From the Future

Rankprop is applicable wherever a relative assessment is more useful or more learnable than an absolute one. One application is domains where quantitative measurements are not available, but relative ones are[8]. For example, a game player might not be able to evaluate moves quantitatively , but might excel at relative move evaluation[10]. Another application is where the goal is to learn to order data drawn from a probability distribution, as in medical risk prediction. But it can also be applied wherever the goal is to order data. For example, in information filtering it is usually important to present more useful information to the user first, not to predict how important each is[5].

MTL is a general method for using related tasks. Here the extra MTL tasks are future measurements. Future measurements are available in many offline learning problems where there is opportunity to collect the measurements for the training set. For example, a robot or autonomous vehicle can more accurately measure the size, location, and identity of objects when it passes near them—road stripes can be detected reliably as a vehicle passes alongside them, but detecting them far ahead of a vehicle is hard. Since driving brings future road into the car's present, stripes can be measured accurately when passed and used as extra features in the training set. They can't be used as *inputs* for learning to drive because they will not be available until too late when driving. As MTL outputs, though, they provide information

that improves learning without requiring they be available at run time[2].

## 8   Summary

This paper presents two methods that can improve generalization on a broad class of problems. This class includes identifying low risk pneumonia patients. The first method, rankprop, tries to learn simple models that support ranking future cases while simultaneously learning to rank the training set. The second, multitask learning, uses lab tests available only during training, as additional target values to bias learning towards a more predictive hidden layer. Experiments using a database of pneumonia patients indicate that together these methods outperform standard backpropagation by 10-50%. Rankprop and MTL are applicable to a large class of problems in which the goal is to learn a relative ranking over the instance space, and where the training data includes features that will not be available at run time. Such problems include identifying higher-risk medical patients as early as possible, identifying lower-risk financial investments, and visual analysis of scenes that become easier to analyze as they are approached in the future.

### Acknowledgements

We thank Greg Cooper, Michael Fine, and other members of the Pitt/CMU Cost-Effective Health Care group for help with the Medis Database. This work was supported by ARPA grant F33615-93-1-1330, NSF grant BES-9315428, Agency for Health Care Policy and Research grant HS06468, and an NSF Graduate Student Fellowship (Baluja).

## Footnotes

[1] Performance at different FOPs sometimes peaks at different epochs. We halt training separately for each FOP in all the experiments to insure this does not confound results.

[2] To make comparisons between methods fair, we first found hidden layer sizes and learning parameters that performed well for each method.

[3] Different representations such as 0.15/0.85 and different error metrics such as cross entropy did not perform better than SSE on 0/1 targets.

## References

[1] Y.S. Abu-Mostafa, "Learning From Hints in Neural Networks," *Journal of Complexity* **6**:2, pp. 192-198, 1989.

[2] R. Caruana, "Learning Many Related Tasks at the Same Time With Backpropagation," *Advances in Neural Information Processing Systems 7*, pp. 656-664, 1995.

[3] R. Caruana, "Multitask Learning: A Knowledge-Based Source of Inductive Bias," *Proceedings of the 10th International Conference on Machine Learning*, pp. 41-48, 1993.

[4] G. Cooper, et al., "An Evaluation of Machine Learning Methods for Predicting Pneumonia Mortality," submitted to *AI in Medicine*, 1995.

[5] K. Lang, "NewsWeeder: Learning to Filter News," *Proceedings of the 12th International Conference on Machine Learning*, pp. 331-339, 1995.

[6] M. Fine, D. Singer, B. Hanusa, J. Lave, and W. Kapoor, "Validation of a Pneumonia Prognostic Index Using the MedisGroups Comparative Hospital Database," *American Journal of Medicine*, **94** 1993.

[7] I. Davis and A. Stentz, "Sensor Fusion For Autonomous Outdoor Navigation Using Neural Networks," *Proceedings of IEEE's Intelligent Robots and Systems Conference*, 1995.

[8] G.T. Hsu, and R. Simmons, "Learning Footfall Evaluation for a Walking Robot," *Proceedings of the 8th International Conference on Machine Learning*, pp. 303-307, 1991.

[9] S.C. Suddarth and A.D.C. Holden, "Symbolic-neural Systems and the Use of Hints for Developing Complex Systems," *International Journal of Man-Machine Studies* **35**:3, pp. 291-311, 1991.

[10] P. Utgoff and S. Saxena, "Learning a Preference Predicate," *Proceedings of the 4th International Conference on Machine Learning*, pp. 115-121, 1987.